# An Information Maximization Approach to Overcomplete and Recurrent Representations

**Oren Shriki and Haim Sompolinsky**
Racah Institute of Physics and
Center for Neural Computation
Hebrew University
Jerusalem, 91904, Israel

**Daniel D. Lee**
Bell Laboratories
Lucent Technologies
Murray Hill, NJ 07974

## Abstract

The principle of maximizing mutual information is applied to learning overcomplete and recurrent representations. The underlying model consists of a network of input units driving a larger number of output units with recurrent interactions. In the limit of zero noise, the network is deterministic and the mutual information can be related to the entropy of the output units. Maximizing this entropy with respect to both the feedforward connections as well as the recurrent interactions results in simple learning rules for both sets of parameters. The conventional independent components (ICA) learning algorithm can be recovered as a special case where there is an equal number of output units and no recurrent connections. The application of these new learning rules is illustrated on a simple two-dimensional input example.

## 1  Introduction

Many unsupervised learning algorithms such as principal component analysis, vector quantization, self-organizing feature maps, and others use the principle of minimizing reconstruction error to learn appropriate features from multivariate data [1, 2]. Independent components analysis (ICA) can similarly be understood as maximizing the likelihood of the data under a non-Gaussian generative model, and thus is related to minimizing a reconstruction cost [3, 4, 5]. On the other hand, the same ICA algorithm can also be derived without regard to a particular generative model by maximizing the mutual information between the data and a nonlinearly transformed version of the data [6]. This principle of information maximization has also been previously applied to explain optimal properties for single units, linear networks, and symplectic transformations [7, 8, 9].

In these proceedings, we show how the principle of maximizing mutual information can be generalized to overcomplete as well as recurrent representations. In the limit of zero noise, we derive gradient descent learning rules for both the feedforward and recurrent weights. Finally, we show the application of these learning rules to some simple illustrative examples.

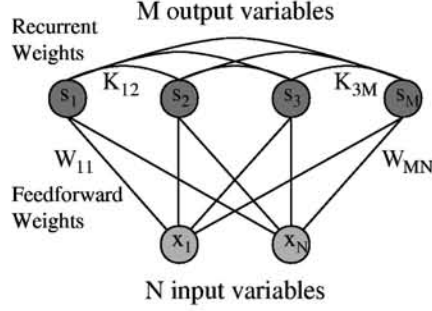

Figure 1: Network diagram of an overcomplete, recurrent representation. $\mathbf{x}$ are input data which influence the output signals $\mathbf{s}$ through feedforward connections $W$. The signals $\mathbf{s}$ also interact with each other through the recurrent interactions $K$.

## 2  Information Maximization

The "Infomax" formulation of ICA considers the problem of maximizing the mutual information between $N$-dimensional data observations $\{\mathbf{x}\}$ which are input to a network resulting in $N$-dimensional output signals $\{\mathbf{s}\}$ [6]. Here, we consider the general problem where the signals $\mathbf{s}$ are $M$-dimensional with $M \geq N$. Thus, the representation is overcomplete because there are more signal components than data components. We also consider the situation where a signal component $s_i$ can influence another component $s_j$ through a recurrent interaction $K_{ji}$. As a network, this is diagrammed in Fig. 1 with the feedforward connections described by the $M \times N$ matrix $W$ and the recurrent connections by the $M \times M$ matrix $K$. The network response $\mathbf{s}$ is a deterministic function of the input $\mathbf{x}$:

$$s_i = g\left(\sum_{j=1}^{N} W_{ij}x_j + \sum_{k=1}^{M} K_{ik}s_k\right) \tag{1}$$

where $g$ is some nonlinear squashing function. In this case, the mutual information between the inputs $\mathbf{x}$ and outputs $\mathbf{s}$ is functionally only dependent on the entropy of the outputs:

$$I(\mathbf{s}, \mathbf{x}) = H(\mathbf{s}) - H(\mathbf{s}|\mathbf{x}) \sim H(\mathbf{s}). \tag{2}$$

The distribution of $\mathbf{s}$ is a $N$-dimensional manifold embedded in a $M$-dimensional vector space and nominally has a negatively divergent entropy. However, as shown in Appendix 1, the probability density of $\mathbf{s}$ can be related to the input distribution via the relation:

$$P(\mathbf{s}) \propto \frac{P(\mathbf{x})}{\sqrt{\det(\chi^T \chi)}} \tag{3}$$

where the susceptibility (or Jacobian) matrix $\chi$ is defined as:

$$\chi_{ij} = \frac{\partial s_i}{\partial x_j}. \tag{4}$$

This result can be understood in terms of the singular value decomposition (SVD) of the matrix $\chi$. The transformation performed by $\chi$ can be decomposed into a series of three transformations: an orthogonal transformation that rotates the axes, a diagonal transformation that scales each axis, followed by another orthogonal transformation. A volume element in the input space is mapped onto a volume element in the output space, and its volume change is described by the diagonal scaling operation. This scale change is given

by the product of the square roots of the eigenvalues of $\chi^T\chi$. Thus, the relationship between the probability distribution in the input and output spaces includes the proportionality factor, $\sqrt{\det(\chi^T\chi)}$, as formally derived in Appendix 1.

We now get the following expression for the entropy of the outputs:

$$H(\mathbf{s}) \sim - \int d\mathbf{x} P(\mathbf{x}) \log\left(\frac{P(\mathbf{x})}{\sqrt{\det(\chi^T\chi)}}\right) = \frac{1}{2}\left\langle \log\det(\chi^T\chi)\right\rangle + H(\mathbf{x}), \qquad (5)$$

where the brackets indicate averaging over the input distribution.

## 3    Learning rules

From Eq. (5), we see that minimizing the following cost function:

$$E = -\frac{1}{2}\mathrm{Tr}\left\langle\log(\chi^T\chi)\right\rangle, \qquad (6)$$

is equivalent to maximizing the mutual information. We first note that the susceptibility $\chi$ satisfies the following recursion relation:

$$\chi_{ij} = g_i' \cdot \left(W_{ij} + \sum_k K_{ik}\chi_{kj}\right) = (GW + GK\chi)_{ij}, \qquad (7)$$

where $G_{ij} = \delta_{ij}g_i'$ and $g_i' \equiv g'\left(\sum_j W_{ij}x_j + \sum_k K_{ik}s_k\right)$.

Solving for $\chi$ in Eq. (7) yields the result:

$$\chi = (G^{-1} - K)^{-1}W = \Phi W, \qquad (8)$$

where $\Phi^{-1} \equiv G^{-1} - K$. $\Phi_{ij}$ can be interpreted as the sensitivity in the recurrent network of the $i$th unit's output to changes in the total input of the $j$th unit.

We next derive the learning rules for the network parameters using gradient descent, as shown in detail in Appendix 2. The resulting expression for the learning rule for the feedforward weights is:

$$\Delta W = -\eta\frac{\partial E}{\partial W} = \eta\left\langle \Gamma^T + \Phi^T\gamma\mathbf{x}^T\right\rangle \qquad (9)$$

where $\eta$ is the learning rate, the matrix $\Gamma$ is defined as

$$\Gamma = (\chi^T\chi)^{-1}\chi^T\Phi \qquad (10)$$

and the vector $\gamma$ is given by

$$\gamma_i = (\chi\Gamma)_{ii}\frac{g_i''}{(g_i')^3}. \qquad (11)$$

Multiplying the gradient in Eq. (9) by the matrix $(WW^T)$ yields an expression analogous to the "natural" gradient learning rule [10]:

$$\Delta W = \eta W\left(I + \left\langle\chi^T\gamma\mathbf{x}^T\right\rangle\right). \qquad (12)$$

Similarly, the learning rule for the recurrent interactions is

$$\Delta K = -\eta\frac{\partial E}{\partial K} = \eta\left\langle(\chi\Gamma)^T + \Phi^T\gamma\mathbf{s}^T\right\rangle. \qquad (13)$$

In the case when there are equal numbers of input and output units, $M = N$, and there are no recurrent interactions, $K = 0$, most of the previous expressions simplify. The susceptibility matrix $\chi$ is diagonal, $\Phi = G$, and $\Gamma = W^{-1}$. Substituting back into Eq. (9) for the learning rule for $W$ results in the update rule:

$$\Delta W = \eta\left[(W^T)^{-1} + \left\langle\mathbf{z}\mathbf{x}^T\right\rangle\right], \qquad (14)$$

where $z_i = g_i''/g_i'$. Thus, the well-known Infomax ICA learning rule is recovered as a special case of Eq. (9) [6].

(a)           (b)           (c)

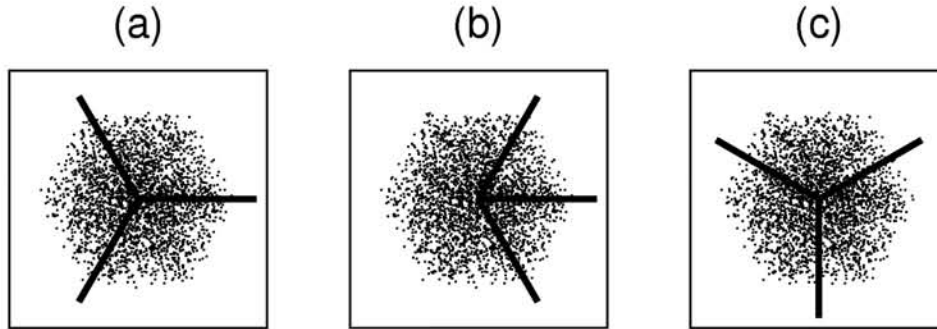

Figure 2: Results of fitting 3 filters to a 2-dimensional hexagon distribution with 10000 sample points.

## 4 Examples

We now apply the preceding learning algorithms to a simple two-dimensional ($N = 2$) input example. Each input point is generated by a linear combination of three (two-dimensional) unit vectors with angles of 0°, 120° and 240°. The coefficients are taken from a uniform distribution on the unit interval. The resulting distribution has the shape of a unit hexagon, which is slightly more dense close to the origin than at the boundaries. Samples of the input distribution are shown in Fig. 2. The second order cross correlations vanish, so that all the structure in the data is described only by higher order correlations. We fix the sigmoidal nonlinearity to be $g(x) = \tanh(x)$.

### 4.1 Feedforward weights

A set of $M = 3$ overcomplete filters for $W$ are learned by applying the update rule in Eq. (9) to random normalized initial conditions while keeping the recurrent interactions fixed at $K = 0$. The length of the rows of $W$ were constrained to be identical so that the filters are projections along certain directions in the two-dimensional space. The algorithm converged after about 20 iterations. Examples of the resulting learned filters are shown by plotting the rows of $W$ as vectors in Fig. 2. As shown in the figure, there are several different local minimum solutions. If the lengths of the rows of $W$ are left unconstrained, slight deviations from these solutions occur, but relative orientation differences of 60° or 120° between the various filters are preserved.

### 4.2 Recurrent interactions

To investigate the effect of recurrent interactions on the representation, we fixed the feedforward weights in $W$ to point in the directions shown in Fig. 2(a), and learned the optimal recurrent interactions $K$ using Eq. (13). Depending upon the length of the rows of $W$ which scaled the input patterns, different optimal values are seen for the recurrent connections. This is shown in Fig. 3 by plotting the value of the cost function against the strength of the uniform recurrent interaction. For small scaled inputs, the optimal recurrent strength is negative which effectively amplifies the output signals since the 3 signals are negatively correlated. With large scaled inputs, the optimal recurrent strength is positive which tend to decrease the outputs. Thus, in this example, optimizing the recurrent connections performs *gain control* on the inputs.

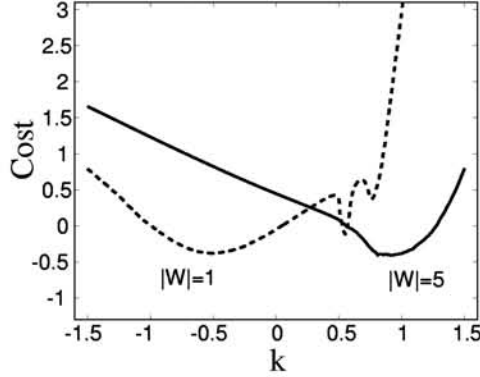

Figure 3: Effect of adding recurrent interactions to the representation. The cost function is plotted as a function of the recurrent interaction strength, for two different input scaling parameters.

## 5  Discussion

The learned feedforward weights are similar to the results of another ICA model that can learn overcomplete representations [11]. Our algorithm, however, does not need to perform approximate inference on a generative model. Instead, it directly maximizes the mutual information between the outputs and inputs of a nonlinear network. Our method also has the advantage of being able to learn recurrent connections that can enhance the representational power of the network. We also note that this approach can be easily generalized to undercomplete representations by simply changing the order of the matrix product in the cost function. However, more work still needs to be done in order to understand technical issues regarding speed of convergence and local minima in larger applications. Possible extensions of this work would be to optimize the nonlinearity that is used, or to adaptively change the number of output units to best match the input distribution.

We acknowledge the financial support of Bell Laboratories, Lucent Technologies, and the US-Israel Binational Science Foundation.

## 6  Appendix 1: Relationship between input and output distributions

In general, the relation between the input and output distributions is given by

$$P(\mathbf{s}) = \int d\mathbf{x} P(\mathbf{x}) P(\mathbf{s}|\mathbf{x}). \tag{15}$$

Since we use a deterministic mapping, the conditional distribution of the response given the input is given by $P(\mathbf{s}|\mathbf{x}) = \delta(\mathbf{s} - g(W\mathbf{x} + K\mathbf{s}))$. By adding independent Gaussian noise to the responses of the output units and considering the limit where the variance of the noise goes to zero, we can write this term as

$$P(\mathbf{s}|\mathbf{x}) = \lim_{\Delta \to 0} \frac{1}{(2\pi\Delta^2)^{N/2}} e^{-\frac{1}{2\Delta^2} \|\mathbf{s} - g(W\mathbf{x} + K\mathbf{s})\|^2}. \tag{16}$$

The output space can be partitioned into those points which belong to the image of the input space, and those which are not. For points outside the image of the input space, $P(\mathbf{s}) = 0$. Consider a point $\mathbf{s}$ inside the image. This means that there exists $\mathbf{x_0}$ such that $\mathbf{s} = g(W\mathbf{x_0} + K\mathbf{s})$. For small $\Delta$, we can expand $g(W\mathbf{x} + K\mathbf{s}) - \mathbf{s} \simeq \chi\delta\mathbf{x}$, where $\chi$ is

defined in Eq. (4), and $\delta\mathbf{x} = \mathbf{x} - \mathbf{x}_0$. We then get

$$
\begin{aligned}
P(\mathbf{s}|\mathbf{x}) &= \lim_{\Delta\to 0}\frac{1}{(2\pi\Delta^2)^{N/2}}e^{-\frac{1}{2}\delta\mathbf{x}^T\left(\frac{\chi^T\chi}{\Delta^2}\right)\delta\mathbf{x}} \\
&= \frac{1}{\sqrt{\det(\chi^T\chi)}}\left[\lim_{\Delta\to 0}\frac{e^{-\frac{1}{2}\delta\mathbf{x}^T\left(\frac{\chi^T\chi}{\Delta^2}\right)\delta\mathbf{x}}}{(2\pi)^{N/2}\sqrt{\det\left(\Delta^2\left(\chi^T\chi\right)^{-1}\right)}}\right].
\end{aligned}
\tag{17}
$$

The expression in the square brackets is a delta function in $\mathbf{x}$ around $\mathbf{x}_0$. Using Eq. (15) we finally get

$$
P(\mathbf{s}) = \frac{P(\mathbf{x})}{\sqrt{\det(\chi^T\chi)}}\Omega(\mathbf{s})
\tag{18}
$$

where the characteristic function $\Omega(\mathbf{s})$ is 1 if $\mathbf{s}$ belongs to the image of the input space and is zero otherwise. Note that for the case when $\chi$ is a square matrix ($M = N$), this expression reduces to the relation $P(\mathbf{s}) = P(\mathbf{x})/|\det(\chi)|$.

# 7  Appendix 2: Derivation of the learning rules

To derive the appropriate learning rules, we need to calculate the derivatives of $E$ with respect to some set of parameters $\lambda$. In general, these derivatives are obtained from the expression:

$$
\frac{\partial E}{\partial\lambda} = -\frac{1}{2}\mathrm{Tr}\left\langle(\chi^T\chi)^{-1}\frac{\partial(\chi^T\chi)}{\partial\lambda}\right\rangle = -\mathrm{Tr}\left\langle(\chi^T\chi)^{-1}\chi^T\frac{\partial\chi}{\partial\lambda}\right\rangle.
\tag{19}
$$

## 7.1  Feedforward weights

In order to derive the learning rule for the weights $W$, we first calculate

$$
\frac{\partial\chi_{ab}}{\partial W_{lm}} = \sum_c\left(\Phi_{ac}\frac{\partial W_{cb}}{\partial W_{lm}} + \frac{\partial\Phi_{ac}}{\partial W_{lm}}W_{cb}\right) = \Phi_{al}\delta_{bm} + \sum_c\frac{\partial\Phi_{ac}}{\partial W_{lm}}W_{cb}.
\tag{20}
$$

From the definition of $\Phi$, we see that:

$$
\frac{\partial\Phi_{ac}}{\partial W_{lm}} = -\sum_{ij}\Phi_{ai}\frac{\partial G_{ij}^{-1}}{\partial W_{lm}}\Phi_{jc}
\tag{21}
$$

and

$$
\frac{\partial G_{ij}^{-1}}{\partial W_{lm}} = -\frac{\delta_{ij}}{(g_i')^2}\frac{\partial g_i'}{\partial W_{lm}} = -\delta_{ij}\frac{g_i''}{(g_i')^3}\frac{\partial s_i}{\partial W_{lm}},
\tag{22}
$$

where $g_i'' \equiv g''\left(\sum_j W_{ij}x_j + \sum_k K_{ik}s_k\right)$.

The derivatives of $\mathbf{s}$ also satisfy a recursion relation similar to Eq. (7):

$$
\frac{\partial s_i}{\partial W_{lm}} = g_i'\cdot\left(\delta_{il}x_m + \sum_j K_{ij}\frac{\partial s_j}{\partial W_{lm}}\right),
\tag{23}
$$

which has the solution:

$$
\frac{\partial s_i}{\partial W_{lm}} = \Phi_{il}x_m.
\tag{24}
$$

Putting all these results together in Eq. (19) and taking the trace, we get the gradient descent rule in Eq. (9).

## 7.2 Recurrent interactions

To derive the learning rules for the recurrent weights $K$, we first calculate the derivatives of $\chi_{ab}$ with respect to $K_{lm}$:

$$\frac{\partial \chi_{ab}}{\partial K_{lm}} = \sum_c \frac{\partial \Phi_{ac}}{\partial K_{lm}} W_{cb} = -\sum_{c,i,j} \Phi_{ai} \frac{\partial \Phi_{ij}^{-1}}{\partial K_{lm}} \Phi_{jc} W_{cb}. \tag{25}$$

From the definition of $\Phi$, we obtain:

$$\frac{\partial \Phi_{ij}^{-1}}{\partial K_{lm}} = -\frac{\delta_{ij}}{(g_i')^2} \frac{\partial g_i'}{\partial K_{lm}} - \delta_{il} \delta_{jm}. \tag{26}$$

The derivatives of $g'$ are obtained from the following relations:

$$\frac{\partial g_i'}{\partial K_{lm}} = \frac{g_i''}{g_i'} \frac{\partial s_i}{\partial K_{lm}} \tag{27}$$

and

$$\frac{\partial s_i}{\partial K_{lm}} = \Phi_{il} s_m. \tag{28}$$

which results from a recursion relation similar to Eq. (23). Finally, after combining these results and calculating the trace, we get the gradient descent learning rule in Eq. (13).

## References

[1] Jolliffe, IT (1986). *Principal Component Analysis.* New York: Springer-Verlag.

[2] Haykin, S (1999). *Neural networks: a comprehensive foundation.* 2nd ed., Prentice-Hall, Upper Saddle River, NJ.

[3] Jutten, C & Herault, J (1991). Blind separation of sources, part I: An adaptive algorithm based on neuromimetic architecture. *Signal Processing* **24**, 1–10.

[4] Hinton, G & Ghahramani, Z (1997). Generative models for discovering sparse distributed representations. *Philosophical Transactions Royal Society B* **352**, 1177–1190.

[5] Pearlmutter, B & Parra, L (1996). A context-sensitive generalization of ICA. In ICONIP'96, 151–157.

[6] Bell, AJ & Sejnowski, TJ (1995). An information maximization approach to blind separation and blind deconvolution. *Neural Comput.* **7**, 1129–1159.

[7] Barlow, HB (1989). Unsupervised learning. *Neural Comput.* **1**, 295–311.

[8] Linsker, R (1992). Local synaptic learning rules suffice to maximize mutual information in a linear network. *Neural Comput.* **4**, 691–702.

[9] Parra, L, Deco, G, & Miesbach, S (1996). Statistical independence and novelty detection with information preserving nonlinear maps. *Neural Comput.* **8**, 260–269.

[10] Amari, S, Cichocki, A & Yang, H (1996). A new learning algorithm for blind signal separation. *Advances in Neural Information Processing Systems* **8**, 757–763.

[11] Lewicki, MS & Sejnowski, TJ (2000). Learning overcomplete representations. *Neural Computation* **12** 337–365.
